# Ensemble Learning
# for Multi-Layer Networks

David Barber*            Christopher M. Bishop[†]

Neural Computing Research Group
Department of Applied Mathematics and Computer Science
Aston University, Birmingham B4 7ET, U.K.
http://www.ncrg.aston.ac.uk/

## Abstract

Bayesian treatments of learning in neural networks are typically
based either on local Gaussian approximations to a mode of the
posterior weight distribution, or on Markov chain Monte Carlo
simulations. A third approach, called *ensemble learning*, was in-
troduced by Hinton and van Camp (1993). It aims to approximate
the posterior distribution by minimizing the Kullback-Leibler di-
vergence between the true posterior and a parametric approximat-
ing distribution. However, the derivation of a deterministic algo-
rithm relied on the use of a Gaussian approximating distribution
with a *diagonal* covariance matrix and so was unable to capture
the posterior correlations between parameters. In this paper, we
show how the ensemble learning approach can be extended to full-
covariance Gaussian distributions while remaining computationally
tractable. We also extend the framework to deal with hyperparam-
eters, leading to a simple re-estimation procedure. Initial results
from a standard benchmark problem are encouraging.

## 1 Introduction

Bayesian techniques have been successfully applied to neural networks in the con-
text of both regression and classification problems (MacKay 1992; Neal 1996). In
contrast to the maximum likelihood approach which finds only a single estimate
for the regression parameters, the Bayesian approach yields a distribution of weight
parameters, $p(\mathbf{w}|D)$, conditional on the training data $D$, and predictions are ex-

Netherlands. http://www.mbfys.kun.nl/snn/ email: davidb@mbfys.kun.nl

[†]Present address: Microsoft Research Limited, St George House, Cambridge CB2 3NH,
UK. http://www.research.microsoft.com email: cmbishop@microsoft.com

pressed in terms of expectations with respect to the posterior distribution (Bishop 1995). However, the corresponding integrals over weight space are analytically intractable. One well-established procedure for approximating these integrals, known as Laplace's method, is to approximate the posterior distribution by a Gaussian, centred at a mode of $p(\mathbf{w}|D)$, in which the covariance of the Gaussian is determined by the local curvature of the posterior distribution (MacKay 1995). The required integrations can then be performed analytically. More recent approaches involve Markov chain Monte Carlo simulations to generate samples from the posterior (Neal 1996). However, such techniques can be computationally expensive, and they also suffer from the lack of a suitable convergence criterion.

A third approach, called ensemble learning, was introduced by Hinton and van Camp (1993) and again involves finding a simple, analytically tractable, approximation to the true posterior distribution. Unlike Laplace's method, however, the approximating distribution is fitted globally, rather than locally, by minimizing a Kullback-Leibler divergence. Hinton and van Camp (1993) showed that, in the case of a Gaussian approximating distribution with a *diagonal* covariance, a deterministic learning algorithm could be derived. Although the approximating distribution is no longer constrained to coincide with a mode of the posterior, the assumption of a diagonal covariance prevents the model from capturing the (often very strong) posterior correlations between the parameters. MacKay (1995) suggested a modification to the algorithm by including linear preprocessing of the inputs to achieve a somewhat richer class of approximating distributions, although this was not implemented. In this paper we show that the ensemble learning approach can be extended to allow a Gaussian approximating distribution with an *general* covariance matrix, while still leading to a tractable algorithm.

## 1.1   The Network Model

We consider a two-layer feed-forward network having a single output whose value is given by

$$f(\mathbf{x}, \mathbf{w}) = \sum_{i=1}^{H} v_i \sigma(\mathbf{u}_i \cdot \mathbf{x}) \tag{1}$$

where $\mathbf{w}$ is a $k$-dimensional vector representing all of the adaptive parameters in the model, $\mathbf{x}$ is the input vector, $\{\mathbf{u}_i\}, i = 1, \ldots, H$ are the input-to-hidden weights, and $\{v_i\}, i = 1, \ldots, H$ are the hidden-to-output weights. The extension to multiple outputs is straightforward. For reasons of analytic tractability, we choose the sigmoidal hidden-unit activation function $\sigma(a)$ to be given by the error function

$$\sigma(a) = \sqrt{\frac{2}{\pi}} \int_0^a \exp\left(-s^2/2\right) ds \tag{2}$$

which (when appropriately scaled) is quantitatively very similar to the standard logistic sigmoid. Hidden unit biases are accounted for by appending the input vector with a node that is always unity. In the current implementation there are no output biases (and the output data is shifted to give zero mean), although the formalism is easily extended to include adaptive output biases (Barber and Bishop 1997). The data set consists of $N$ pairs of input vectors and corresponding target output values $D = \{\mathbf{x}^\mu, t^\mu\}, \mu = 1, \ldots, N$. We make the standard assumption of Gaussian noise on the target values, with variance $\beta^{-1}$. The likelihood of the training data is then proportional to $\exp(-\beta E_D)$, where the training error $E_D$ is

$$E_D(\mathbf{w}) = \frac{1}{2} \sum_\mu \left(f(\mathbf{x}^\mu, \mathbf{w}) - t^\mu\right)^2. \tag{3}$$

The prior distribution over weights is chosen to be a Gaussian of the form

$$p(\mathbf{w}) \propto \exp\left(-E_w(\mathbf{w})\right) \tag{4}$$

where $E_w(\mathbf{w}) = \frac{1}{2}\mathbf{w}^T\mathbf{A}\mathbf{w}$, and $\mathbf{A}$ is a matrix of hyperparameters. The treatment of $\beta$ and $\mathbf{A}$ is dealt with in Section 2.1. From Bayes' theorem, the posterior distribution over weights can then be written

$$p(\mathbf{w}|D) = \frac{1}{Z}\exp\left(-\beta E_D(\mathbf{w}) - E_w(\mathbf{w})\right) \tag{5}$$

where $Z$ is a normalizing constant. Network predictions on a novel example are given by the posterior average of the network output

$$\langle f(\mathbf{x})\rangle = \int f(\mathbf{x},\mathbf{w})p(\mathbf{w}|D)\,d\mathbf{w}. \tag{6}$$

This represents an integration over a high-dimensional space, weighted by a posterior distribution $p(\mathbf{w}|D)$ which is exponentially small except in narrow regions whose locations are unknown a-priori. The accurate evaluation of such integrals is thus very difficult.

## 2 Ensemble Learning

Integrals of the form (6) may be tackled by approximating $p(\mathbf{w}|D)$ by a simpler distribution $Q(\mathbf{w})$. In this paper we choose this approximating distribution to be a Gaussian with mean $\overline{w}$ and covariance $\mathbf{C}$. We determine the values of $\overline{w}$ and $\mathbf{C}$ by minimizing the Kullback-Leibler divergence between the network posterior and approximating Gaussian, given by

$$\begin{aligned} \mathcal{F}[Q] &= \int Q(\mathbf{w})\ln\left\{\frac{Q(\mathbf{w})}{p(\mathbf{w}|D)}\right\}d\mathbf{w} &(7)\\ &= \int Q(\mathbf{w})\ln Q(\mathbf{w})d\mathbf{w} - \int Q(\mathbf{w})\ln p(\mathbf{w}|D)\,d\mathbf{w}. &(8) \end{aligned}$$

The first term in (8) is the negative entropy of a Gaussian distribution, and is easily evaluated to give $\frac{1}{2}\ln\det(\mathbf{C}) + \text{const.}$

From (5) we see that the posterior dependent term in (8) contains two parts that depend on the prior and likelihood

$$\int Q(\mathbf{w})E_w(\mathbf{w})d\mathbf{w} + \int Q(\mathbf{w})E_D(\mathbf{w})d\mathbf{w}. \tag{9}$$

Note that the normalization coefficient $Z^{-1}$ in (5) gives rise to a constant additive term in the KL divergence and so can be neglected. The prior term $E_w(\mathbf{w})$ is quadratic in $\mathbf{w}$, and integrates to give $\text{Tr}(\mathbf{C}\mathbf{A}) + \frac{1}{2}\overline{w}^T\mathbf{A}\overline{w}$. This leaves the data dependent term in (9) which we write as

$$L = \int Q(\mathbf{w})E_D(\mathbf{w})dw = \frac{\beta}{2}\sum_{\mu=1}^{N}l(\mathbf{x}^\mu, t^\mu) \tag{10}$$

where

$$l(\mathbf{x},t) = \int Q(\mathbf{w})\left(f(\mathbf{x},\mathbf{w})\right)^2 d\mathbf{w} - 2t\int Q(\mathbf{w})f(\mathbf{x},\mathbf{w})\,d\mathbf{w} + t^2. \tag{11}$$

For clarity, we concentrate only on the first term in (11), as the calculation of the term linear in $f(\mathbf{x}, \mathbf{w})$ is similar, though simpler. Writing the Gaussian integral over $Q$ as an average, $\langle\ \rangle$, the first term of (11) becomes

$$\left\langle \left(f(\mathbf{x}, \mathbf{w})\right)^2 \right\rangle = \sum_{i,j=1}^{H} \left\langle v_i v_j \sigma(\mathbf{u}_i^T \mathbf{x}) \sigma(\mathbf{u}_j^T \mathbf{x}) \right\rangle. \tag{12}$$

To simplify the notation, we denote the set of input-to-hidden weights $(\mathbf{u}_1, \ldots, \mathbf{u}_H)$ by $\mathbf{u}$ and the set of hidden-to-output weights, $(v_1, \ldots, v_H)$ by $\mathbf{v}$. Similarly, we partition the covariance matrix $\mathbf{C}$ into blocks, $\mathbf{C}_{uu}$, $\mathbf{C}_{vu}$, $\mathbf{C}_{vv}$, and $\mathbf{C}_{vu} = \mathbf{C}_{uv}^T$. As the components of $\mathbf{v}$ do not enter the non-linear sigmoid functions, we can directly integrate over $\mathbf{v}$, so that each term in the summation (12) gives

$$\left\langle \left(\theta_{ij} + (\mathbf{u} - \overline{\mathbf{u}})^T \Psi_{ij} (\mathbf{u} - \overline{\mathbf{u}}) + \Omega_{ij}^T (\mathbf{u} - \overline{\mathbf{u}})\right) \sigma\left(\mathbf{u}^T \mathbf{x}^i\right) \sigma\left(\mathbf{u}^T \mathbf{x}^j\right) \right\rangle \tag{13}$$

where

$$\theta_{ij} = \left(\mathbf{C}_{vv} - \mathbf{C}_{vu}\mathbf{C}_{uu}^{-1}\mathbf{C}_{uv}\right)_{ij} + \overline{v}_i \overline{v}_j \tag{14}$$

$$\Psi_{ij} = \mathbf{C}_{uu}^{-1}\mathbf{C}_{u,v=i}\mathbf{C}_{v=j,u}\mathbf{C}_{uu}^{-1}, \tag{15}$$

$$\Omega_{ij} = 2\mathbf{C}_{uu}^{-1}\mathbf{C}_{u,v=j}\overline{v}_i. \tag{16}$$

Although the remaining integration in (13) over $\mathbf{u}$ is not analytically tractable, we can make use of the following result to reduce it to a one-dimensional integration

$$\left\langle \sigma\left(\mathbf{z}\cdot\mathbf{a} + a_0\right) \sigma\left(\mathbf{z}\cdot\mathbf{b} + b_0\right) \right\rangle_{\mathbf{z}} = \left\langle \sigma\left(z|\mathbf{a}| + a_0\right) \sigma\left(\frac{z\mathbf{a}^T\mathbf{b} + b_0|\mathbf{a}|}{\sqrt{|\mathbf{a}|^2 \left(1 + |\mathbf{b}|^2\right) - \left(\mathbf{a}^T\mathbf{b}\right)^2}}\right) \right\rangle_{z} \tag{17}$$

where $\mathbf{a}$ and $\mathbf{b}$ are vectors and $a_0, b_0$ are scalar offsets. The average on the left of (17) is over an isotropic multi-dimensional Gaussian, $p(\mathbf{z}) \propto \exp(-\mathbf{z}^T\mathbf{z}/2)$, while the average on the right is over the one-dimensional Gaussian $p(z) \propto \exp(-z^2/2)$. This result follows from the fact that the vector $\mathbf{z}$ only occurs through the scalar product with $\mathbf{a}$ and $\mathbf{b}$, and so we can choose a coordinate system in which the first two components of $\mathbf{z}$ lie in the plane spanned by $\mathbf{a}$ and $\mathbf{b}$. All orthogonal components do not appear elsewhere in the integrand, and therefore integrate to unity.

The integral we desire, (13) is only a little more complicated than (17) and can be evaluated by first transforming the coordinate system to an isotopic basis $\mathbf{z}$, and then differentiating with respect to elements of the covariance matrix to 'pull down' the required linear and quadratic terms in the $\sigma$-independent pre-factor of (13). These derivatives can then be reduced to a form which requires only the numerical evaluation of (17). We have therefore succeeded in reducing the calculation of the KL divergence to analytic terms together with a single one-dimensional numerical integration of the form (17), which we compute using Gaussian quadrature[1].

Similar techniques can be used to evaluate the derivatives of the KL divergence with respect to the mean and covariance matrix (Barber and Bishop 1997). Together with the KL divergence, these derivatives are then used in a scaled conjugate gradient optimizer to find the parameters $\overline{\mathbf{w}}$ and $\mathbf{C}$ that represent the best Gaussian fit.

The number of parameters in the covariance matrix scales quadratically with the number of weight parameters. We therefore have also implemented a version with

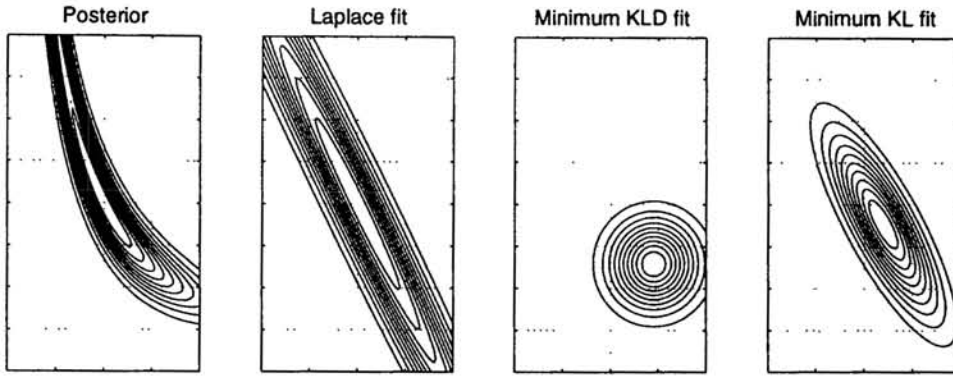

Figure 1: Laplace and minimum Kullback-Leibler Gaussian fits to the posterior. The Laplace method underestimates the local posterior mass by basing the covariance matrix on the mode alone, and has KL value 41. The minimum Kullback-Leibler Gaussian fit with a diagonal covariance matrix (KLD) gives a KL value of 4.6, while the minimum Kullback-Leibler Gaussian with full covariance matrix achieves a value of 3.9.

a constrained covariance matrix

$$\mathbf{C} = \text{diag}(d_1^2, \ldots, d_n^2) + \sum_{i=1}^{s} \mathbf{s}_i \mathbf{s}_i^T \tag{18}$$

which is the form of covariance used in factor analysis (Bishop 1997). This reduces the number of free parameters in the covariance matrix from $k(k+1)/2$ to $k(s+1)$ (representing $k(s+1) - s(s-1)/2$ independent degrees of freedom) which is now linear in $k$. Thus, the number of parameters can be controlled by changing $s$ and, unlike a diagonal covariance matrix, this model can still capture the strongest of the posterior correlations. The value of $s$ should be as large as possible, subject only to computational cost limitations. There is no 'over-fitting' as $s$ is increased since more flexible distributions $Q(\mathbf{w})$ simply better approximate the true posterior.

We illustrate the optimization of the KL divergence using a toy problem involving the posterior distribution for a two-parameter regression problem. Figure 1 shows the true posterior together with approximations obtained from Laplace's method, ensemble learning with a diagonal covariance Gaussian, and ensemble learning using an unconstrained Gaussian.

## 2.1 Hyperparameter Adaptation

So far, we have treated the hyperparameters as fixed. We now extend the ensemble learning formalism to include hyperparameters within the Bayesian framework. For simplicity, we consider a standard isotropic prior covariance matrix of the form $\mathbf{A} = \alpha\mathbf{I}$, and introduce hyperpriors given by Gamma distributions

$$\ln p(\alpha) = \ln\left\{\alpha^{a-1}\exp\left(-\frac{\alpha}{b}\right)\right\} + \text{const} \tag{19}$$

$$\ln p(\beta) = \ln\left\{\beta^{c-1}\exp\left(-\frac{\beta}{d}\right)\right\} + \text{const} \tag{20}$$

where $a, b, c, d$ are constants. The joint posterior distribution of the weights and hyperparameters is given by

$$p(\mathbf{w}, \alpha, \beta | D) \propto p(D|\mathbf{w}, \beta) \, p(\mathbf{w}|\alpha) \, p(\alpha) \, p(\beta) \qquad (21)$$

in which

$$\ln p(D|\mathbf{w}, \beta) = -\beta E_D + \frac{N}{2} \ln \beta + \text{const} \qquad (22)$$

$$\ln p(\mathbf{w}|\alpha) = -\alpha|\mathbf{w}|^2 + \frac{k}{2} \ln \alpha + \text{const} \qquad (23)$$

We follow MacKay (1995) by modelling the joint posterior $p(\mathbf{w}, \alpha, \beta | D)$ by a factorized approximating distribution of the form

$$Q(\mathbf{w})R(\alpha)S(\beta) \qquad (24)$$

where $Q(\mathbf{w})$ is a Gaussian distribution as before, and the functional forms of $R$ and $S$ are left unspecified. We then minimize the KL divergence

$$\mathcal{F}[Q, R, S] = \int Q(\mathbf{w})R(\alpha)S(\beta) \ln \left\{ \frac{Q(\mathbf{w})R(\alpha)S(\beta)}{p(\mathbf{w}, \alpha, \beta|D)} \right\} \, d\mathbf{w} \, d\alpha \, d\beta. \qquad (25)$$

Consider first the dependence of (25) on $Q(\mathbf{w})$

$$\mathcal{F}[Q] = -\int Q(\mathbf{w})R(\alpha)S(\beta) \left\{ -\beta E_D(\mathbf{w}) - \frac{\alpha}{2}|\mathbf{w}|^2 - \ln Q(\mathbf{w}) \right\} + \text{const} \qquad (26)$$

$$= -\int Q(\mathbf{w}) \left\{ -\overline{\beta} E_D(\mathbf{w}) - \frac{\overline{\alpha}}{2}|\mathbf{w}|^2 - \ln Q(\mathbf{w}) \right\} + \text{const} \qquad (27)$$

where $\overline{\alpha} = \int R(\alpha)\alpha d\alpha$ and $\overline{\beta} = \int S(\beta)\beta d\beta$. We see that (27) has the form of (8), except that the fixed hyperparameters are now replaced with their average values. To calculate these averages, consider the dependence of the functional $\mathcal{F}$ on $R(\alpha)$

$$\mathcal{F}[R] = -\int Q(\mathbf{w})R(\alpha)S(\beta) \left\{ -\frac{\alpha}{2}|\mathbf{w}|^2 + \frac{k}{2} \ln \alpha + (a-1) \ln \alpha - \frac{\alpha}{b} \right\} \, d\mathbf{w} \, d\alpha \, d\beta$$

$$= -\int R(\alpha) \left\{ \frac{\alpha}{s} + (r-1) \ln \alpha - \ln R(\alpha) \right\} \, d\alpha + \text{const} \qquad (28)$$

where $r = \frac{k}{2} + a$ and $1/s = \frac{1}{2}|\overline{\mathbf{w}}|^2 + \frac{1}{2}\text{Tr}\mathbf{C} + 1/b$. We recognise (28) as the Kullback-Leibler divergence between $R(\alpha)$ and a Gamma distribution. Thus the optimum $R(\alpha)$ is also Gamma distributed

$$R(\alpha) \propto \alpha^{r-1} \exp \left( -\frac{\alpha}{s} \right). \qquad (29)$$

We therefore obtain $\overline{\alpha} = rs$.

A similar procedure for $S(\beta)$ gives $\overline{\beta} = uv$, where $u = \frac{N}{2} + c$ and $1/v = \langle E_D \rangle + 1/d$, in which $\langle E_D \rangle$ has already been calculated during the optimization of $Q(\mathbf{w})$.

This defines an iterative procedure in which we start by initializing the hyperparameters (using the mean of the hyperprior distributions) and then alternately optimize the KL divergence over $Q(\mathbf{w})$ and re-estimate $\overline{\alpha}$ and $\overline{\beta}$.

## 3  Results and Discussion

As a preliminary test of our method on a standard benchmark problem, we applied the minimum KL procedure to the Boston Housing dataset. This is a one

| Method | Test Error |
|---|---|
| Ensemble ($s = 1$) | 0.22 |
| Ensemble (diagonal) | 0.28 |
| Laplace | 0.33 |

Table 1: Comparison of ensemble learning with Laplace's method. The test error is defined to be the mean squared error over the test set of 378 examples.

dimensional regression problem, with 13 inputs, in which the data for 128 training examples was obtained from the DELVE archive[2]. We trained a network of four hidden units, with covariance matrix given by (18) with $s = 1$, and specified broad hyperpriors on $\alpha$ and $\beta$ ($a = 0.25$, $b = 400$, $c = 0.05$, and $d = 2000$). Predictions are made by evaluating the integral in (6). This integration can be done analytically as a consequence of the form of the sigmoid function given in (2).

We compared the performance of the KL method against the Laplace framework of MacKay (1995) which also treats hyperparameters through a re-estimation procedure. In addition we also evaluated the performance of the ensemble method using a diagonal covariance matrix. Our results are summarized in Table 1.

**Acknowledgements**

We would like to thank Chris Williams for helpful discussions. Supported by EPSRC grant GR/J75425: *Novel Developments in Learning Theory for Neural Networks*.

## Footnotes

*Present address: SNN, University of Nijmegen, Geert Grooteplein 21, Nijmegen, The

[1] Although (17) appears to depend on 4 parameters, it can be expressed in terms of 3 independent parameters. An alternative to performing quadrature during training would therefore be to compute a 3-dimensional look-up table in advance.

[2]See http://www.cs.utoronto.ca/~delve/

# References

Barber, D. and C. M. Bishop (1997). On computing the KL divergence for Bayesian neural networks. Technical report, Neural Computing Research Group, Aston University, Birmingham, U.K.

Bishop, C. M. (1995). *Neural Networks for Pattern Recognition*. Oxford University Press.

Bishop, C. M. (1997). Latent variables, mixture distributions and topographic mappings. Technical report, Aston University. To appear in *Proceedings of the NATO Advanced Study Institute on Learning in Graphical Models*, Erice.

Hinton, G. E. and D. van Camp (1993). Keeping neural networks simple by minimizing the description length of the weights. In *Proceedings of the Sixth Annual Conference on Computational Learning Theory*, pp. 5–13.

MacKay, D. J. C. (1992). A practical Bayesian framework for back-propagation networks. *Neural Computation 4*(3), 448–472.

MacKay, D. J. C. (1995). Developments in probabilistic modelling with neural networks—ensemble learning. In *Neural Networks: Artificial Intelligence and Industrial Applications. Proceedings of the 3rd Annual Symposium on Neural Networks, Nijmegen, Netherlands, 14-15 September 1995*, Berlin, pp. 191–198. Springer.

MacKay, D. J. C. (1995). Probable networks and plausible predictions – a review of practical Bayesian methods for supervised neural networks. *Network: Computation in Neural Systems 6*(3), 469–505.

Neal, R. M. (1996). *Bayesian Learning for Neural Networks*. Springer. Lecture Notes in Statistics 118.

